# Invariant Pattern Recognition
# by Semidefinite Programming Machines

**Thore Graepel**
Microsoft Research Ltd.
Cambridge, UK
*thoreg@microsoft.com*

**Ralf Herbrich**
Microsoft Research Ltd.
Cambridge, UK
*rherb@microsoft.com*

## Abstract

Knowledge about local invariances with respect to given pattern transformations can greatly improve the accuracy of classification. Previous approaches are either based on regularisation or on the generation of virtual (transformed) examples. We develop a new framework for learning linear classifiers under known transformations based on semidefinite programming. We present a new learning algorithm—the Semidefinite Programming Machine (SDPM)—which is able to find a maximum margin hyperplane when the training examples are polynomial trajectories instead of single points. The solution is found to be sparse in dual variables and allows to identify those points on the trajectory with minimal real-valued output as virtual support vectors. Extensions to segments of trajectories, to more than one transformation parameter, and to learning with kernels are discussed. In experiments we use a Taylor expansion to locally approximate rotational invariance in pixel images from USPS and find improvements over known methods.

## 1   Introduction

One of the central problems of pattern recognition is the exploitation of known invariances in the pattern domain. In images these invariances may include rotation, translation, shearing, scaling, brightness, and lighting direction. In addition, specific domains such as handwritten digit recognition may exhibit invariances such as line thinning/thickening and other non-uniform deformations [8]. The challenge is to combine the training sample with the knowledge of invariances to obtain a good classifier.

Possibly the most straightforward way of incorporating invariances is by including virtual examples into the training sample which have been generated from actual examples by the application of the invariance $\mathbf{T} : \mathbb{R} \times \mathbb{R}^n \to \mathbb{R}^n$ at some fixed $\theta \in \mathbb{R}$, e.g. the method of virtual support vectors [7]. Images $\mathbf{x}$ subjected to the transformation $\mathbf{T}(\theta, \cdot)$ describe highly non-linear trajectories or manifolds in pixel space. The tangent distance [8] approximates the distance between the trajectories (manifolds) by the distance between their tangent vectors (planes) at a given value $\theta = \theta_0$ and can be used with any kind of distance-based classifier. Another approach, tangent prop [8], incorporates the invariance $\mathbf{T}$ directly into the objective function for learning by penalising large values of the derivative of the classification function w.r.t. the given

transformation parameter. A similar regulariser can be applied to support vector machines [1].

We take up the idea of considering the trajectory given by the combination of training vector and transformation. While data in machine learning are commonly represented as vectors $\mathbf{x} \in \mathbb{R}^n$ we instead consider more complex training examples each of which is represented as a (usually infinite) set

$$\{\mathbf{T}(\theta, \mathbf{x}_i) \,:\, \theta \in \mathbb{R}\} \subset \mathbb{R}^n, \tag{1}$$

which constitutes a trajectory in $\mathbb{R}^n$. Our goal is to learn a linear classifier that separates well the training trajectories belonging to different classes. In practice, we may be given a "standard" training example $\mathbf{x}$ together with a differentiable transformation $\mathbf{T}$ representing an invariance of the learning problem. The problem can be solved if the transformation $\mathbf{T}$ is approximated by a transformation $\tilde{\mathbf{T}}$ polynomial in $\theta$, e.g., a Taylor expansion of the form

$$\tilde{\mathbf{T}}(\theta, \mathbf{x}_i) \approx \sum_{j=0}^{r} \theta^j \cdot \left( \frac{1}{j!} \left. \frac{d^j \mathbf{T}(\theta, \mathbf{x}_i)}{d\theta^j} \right|_{\theta=0} \right) = \sum_{j=0}^{r} \theta^j \cdot (\mathbf{X}_i)_{j,\cdot}. \tag{2}$$

Our approach is based on a powerful theorem by Nesterov [5] which states that the set $\mathcal{P}_{2l}^+$ of polynomials of degree $2l$ non-negative on the entire real line is a convex set representable by positive semidefinite (psd) constraints. Hence, optimisation over $\mathcal{P}_{2l}^+$ can be formulated as a semidefinite program (SDP). Recall that an SDP [9] is given by a linear objective function minimised subject to a linear matrix inequality (LMI),

$$\underset{\mathbf{w} \in \mathbb{R}^n}{\text{minimise}} \ \ \mathbf{c}^\top \mathbf{w} \quad \text{subject to} \quad \mathbf{A}(\mathbf{w}) := \sum_{j=1}^{n} w_j \mathbf{A}_j - \mathbf{B} \succeq \mathbf{0}, \tag{3}$$

with $\mathbf{A}_j \in \mathbb{R}^{m \times m}$ for all $j \in \{0, \ldots, n\}$. The LMI $\mathbf{A}(\mathbf{w}) \succeq \mathbf{0}$ means that $\mathbf{A}(\mathbf{w})$ is required to be positive semidefinite, i.e., that for all $\mathbf{v} \in \mathbb{R}^n$ we have $\mathbf{v}^\top \mathbf{A}(\mathbf{w}) \mathbf{v} = \sum_{j=1}^{n} w_j (\mathbf{v}^\top \mathbf{A}_j \mathbf{v}) - \mathbf{v}^\top \mathbf{B} \mathbf{v} \geq 0$ which reveals that LMI constraints correspond to infinitely many linear constraints. This expressive power can be used to enforce constraints for training examples as given by (1), i.e., constraints required to hold for all values $\theta \in \mathbb{R}$. Based on this representability theorem for non-negative polynomials we develop a learning algorithm—the Semidefinite Programming Machine (SDPM)—that maximises the margin on polynomial training samples, much like the support vector machine [2] for ordinary single vector data.

## 2 Semidefinite Programming Machines

**Linear Classifiers and Polynomial Examples** We consider binary classification problems and linear classifiers. Given a training sample $((\mathbf{x}_1, y_1), \ldots, (\mathbf{x}_m, y_m)) \in (\mathbb{R}^n \times \{-1, +1\})^m$ we aim at learning a weight vector[1] $\mathbf{w} \in \mathbb{R}^n$ to classify examples $\mathbf{x}$ by $y(\mathbf{x}) = \text{sign}(\mathbf{w}^\top \mathbf{x})$. Assuming linear separability of the training sample the principle of empirical risk minimisation recommends finding a weight vector $\mathbf{w}$ such that for all $i \in \{1, \ldots, m\}$ we have $y_i \mathbf{w}^\top \mathbf{x}_i \geq 0$. As such this constitutes a linear feasibility problem and is easily solved by the perceptron algorithm [6]. Additionally requiring the solution to maximise the margin leads to the well-known quadratic program of support vector learning [2].

In order to be able to cope with known invariances $\mathbf{T}(\theta, \cdot)$ we would like to generalise the above setting to the following feasibility problem:

$$\text{find } \mathbf{w} \in \mathbb{R}^n \quad \text{such that} \quad \forall i \in \{1, \ldots, m\} : \forall \theta \in \mathbb{R} : \quad y_i \mathbf{w}^\top \mathbf{x}_i(\theta) \geq 0, \tag{4}$$

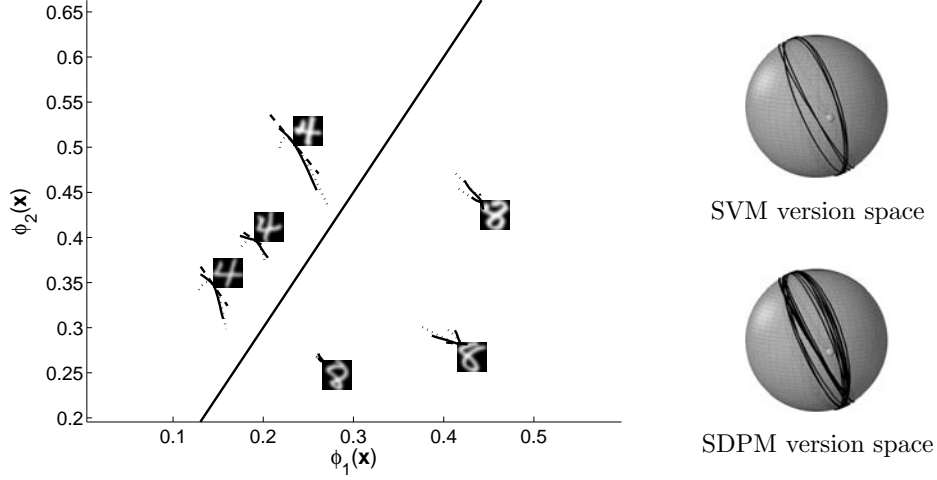

Figure 1: (**Left**) Approximated trajectories for rotated USPS images (2) for $r = 1$ (dashed line) and $r = 2$ (dotted line). The features are the mean pixel intensities in the top and bottom half of the image. (**Right**) Set of weight vectors $\mathbf{w}$ which are consistent with the six images (top) and the six trajectories (bottom). The SDPM version space is smaller and thus determines the weight vector more precisely. The dot corresponds to the separating plane in the left plot.

that is, we would require the weight vector to classify correctly every transformed training example $\mathbf{x}_i(\theta) := \mathbf{T}(\theta, \mathbf{x}_i)$ for every value of the transformation parameter $\theta$. The situation is illustrated in Figure 1. In general, such a set of constraints leads to a very complex and difficult-to-solve feasibility problem. As a consequence, we consider only transformations $\tilde{\mathbf{T}}(\theta, \mathbf{x})$ of polynomial form, i.e., $\tilde{\mathbf{x}}_i(\theta) := \tilde{\mathbf{T}}(\theta, \mathbf{x}_i) = \mathbf{X}_i^\top \boldsymbol{\theta}$, each polynomial example $\tilde{\mathbf{x}}_i(\theta)$ being represented by a polynomial in the row vectors of $\mathbf{X}_i \in \mathbb{R}^{(r+1)\times n}$, with $\boldsymbol{\theta} := (1, \theta, \ldots, \theta^r)^\top$. Then the problem (4) can be written as

$$\text{find } \mathbf{w} \in \mathbb{R}^n \quad \text{such that} \quad \forall i \in \{1, \ldots, m\} : \forall \theta \in \mathbb{R} : \quad y_i \mathbf{w}^\top \mathbf{X}_i^\top \boldsymbol{\theta} \geq 0, \quad (5)$$

which is equivalent to finding a weight vector $\mathbf{w}$ such that the polynomials $p_i(\theta) := y_i \mathbf{w}^\top \mathbf{X}_i^\top \boldsymbol{\theta}$ are non-negative everywhere, i.e., $p_i \in \mathcal{P}_r^+$. The following proposition by Nesterov [5] paves the way for an SDP formulation of the above problem if $r = 2l$.

**Proposition 1 (SD Representation of Non-Negative Polynomials [5]).** *The set $\mathcal{P}_{2l}^+$ of polynomials non-negative everywhere on the real line is SD-representable:*

  *1. For every $\mathbf{P} \succeq \mathbf{0}$ the polynomial $p(\theta) = \boldsymbol{\theta}^\top \mathbf{P} \boldsymbol{\theta}$ is non-negative everywhere.*

  *2. For every polynomial $p \in \mathcal{P}_{2l}^+$ there exists a $\mathbf{P} \succeq \mathbf{0}$ such that $p(\theta) = \boldsymbol{\theta}^\top \mathbf{P} \boldsymbol{\theta}$.*

*Proof.* Any polynomial $p \in \mathcal{P}_{2l}$ can be written as $p(\theta) = \boldsymbol{\theta}^\top \mathbf{P} \boldsymbol{\theta}$, where $\mathbf{P} = \mathbf{P}^\top \in \mathbb{R}^{(l+1)\times(l+1)}$. *Statement 1*: $\mathbf{P} \succeq \mathbf{0}$ implies $\forall \theta \in \mathbb{R} : p(\theta) = \boldsymbol{\theta}^\top \mathbf{P} \boldsymbol{\theta} = \|\mathbf{P}^{\frac{1}{2}}\boldsymbol{\theta}\|^2 \geq 0$, hence $p \in \mathcal{P}_{2l}^+$. *Statement 2*: Every non-negative polynomial $p \in \mathcal{P}_{2l}^+$ can be written as a sum of squared polynomials [4], hence $\exists q_i \in \mathcal{P}_l : p(\theta) = \sum_i q_i^2(\theta) = \boldsymbol{\theta}^\top \left( \sum_i \mathbf{q}_i \mathbf{q}_i^\top \right) \boldsymbol{\theta}$ where $\mathbf{P} := \sum_i \mathbf{q}_i \mathbf{q}_i^\top \succeq \mathbf{0}$ and $\mathbf{q}_i$ is the coefficient vector of polynomial $q_i$. $\qquad \square$

**Maximising Margins on Polynomial Samples** Here we develop an SDP formulation for learning a maximum margin classifier given the polynomial constraints

(5). It is well-known that SDPs include quadratic programs as a special case [9]. The squared objective $\|\mathbf{w}\|^2$ is minimised by replacing it with an auxiliary variable $t$ subject to a quadratic constraint $t \geq \|\mathbf{w}\|^2$ that is written as an LMI using Schur's complement lemma,

$$\underset{(\mathbf{w},t)}{\text{minimise}} \quad \frac{1}{2}t \quad \text{subject to} \quad \mathbf{F}(\mathbf{w},t) := \left( \begin{array}{cc} \mathbf{I}_n & \mathbf{w} \\ \mathbf{w}^\top & t \end{array} \right) \succeq \mathbf{0},$$

$$\text{and} \ \forall i: \ \mathbf{G}(\mathbf{w}, \mathbf{X}_i, y_i) := \mathbf{G}_0 + \sum_{j=1}^{n} w_j \mathbf{G}_j \left( (\mathbf{X}_i)_{.,j}, y_i \right) \succeq \mathbf{0}. \quad (6)$$

This constitutes an SDP as in (3) by the fact that a block-diagonal matrix is psd if and only if all its diagonal blocks are psd.

For the sake of illustration consider the case of $l = 0$ (the simplest non-trivial case). The matrix $\mathbf{G}(\mathbf{w}, \mathbf{X}_i, y_i)$ reduces to a scalar $y_i \mathbf{w}^\top \mathbf{x}_i - 1$, which translates into the standard SVM constraint $y_i \mathbf{w}^\top \mathbf{x}_i \geq 1$ linear in $\mathbf{w}$.

For the case $l = 1$ we have $\mathbf{G}(\mathbf{w}, \mathbf{X}_i, y_i) \in \mathbb{R}^{2 \times 2}$ and

$$\mathbf{G}(\mathbf{w}, \mathbf{X}_i, y_i) = \left( \begin{array}{cc} y_i \mathbf{w}^\top (\mathbf{X}_i)_{0,.} - 1 & \frac{1}{2} y_i \mathbf{w}^\top (\mathbf{X}_i)_{1,.} \\ \frac{1}{2} y_i \mathbf{w}^\top (\mathbf{X}_i)_{1,.} & y_i \mathbf{w}^\top (\mathbf{X}_i)_{2,.} \end{array} \right). \quad (7)$$

Although we require $\mathbf{G}(\mathbf{w}, \mathbf{X}_i, y_i)$ to be psd the resulting optimisation problem can be formulated in terms of a second-order cone program (SOCP) because the matrices involved are only $2 \times 2$.[2]

For the case $l \geq 2$ the resulting program constitutes a genuine SDP. Again for the sake of illustration we consider the case $l = 2$ first. Since a polynomial $p$ of degree four is fully determined by its five coefficients $p_0, \ldots, p_4$, but the symmetric matrix $\mathbf{P} \in \mathbb{R}^{3 \times 3}$ in $p(\theta) = \boldsymbol{\theta}^\top \mathbf{P} \boldsymbol{\theta}$ has six degrees of freedom we require one auxiliary variable $u_i$ per training example,

$$\mathbf{G}(\mathbf{w}, u_i, \mathbf{X}_i, y_i) = \frac{1}{2} \left( \begin{array}{ccc} 2 y_i \mathbf{w}^\top (\mathbf{X}_i)_{0,.} - 2 & y_i \mathbf{w}^\top (\mathbf{X}_i)_{1,.} & y_i \mathbf{w}^\top (\mathbf{X}_i)_{2,.} - u_i \\ y_i \mathbf{w}^\top (\mathbf{X}_i)_{1,.} & 2u_i & y_i \mathbf{w}^\top (\mathbf{X}_i)_{3,.} \\ y_i \mathbf{w}^\top (\mathbf{X}_i)_{2,.} - u_i & y_i \mathbf{w}^\top (\mathbf{X}_i)_{3,.} & y_i \mathbf{w}^\top (\mathbf{X}_i)_{4,.} \end{array} \right).$$

In general, since a polynomial of degree $2l$ has $2l + 1$ coefficients and a symmetric $(l+1) \times (l+1)$ matrix has $(l+1)(l+2)/2$ degrees of freedom we require $(l-1)l/2$ auxiliary variables.

**Dual Program and Complementarity** Let us consider the dual SDPs corresponding to the optimisation problems above. For the sake of clarity, we restrict the presentation to the case $l = 1$. The dual of the general SDP (3) is given by

$$\underset{\boldsymbol{\Lambda} \in \mathbb{R}^{m \times m}}{\text{maximise}} \ \text{tr}(\mathbf{B}\boldsymbol{\Lambda}) \quad \text{subject to} \quad \forall j \in \{1, \ldots, n\}: \ \text{tr}(\mathbf{A}_j \boldsymbol{\Lambda}) = c_j; \ \boldsymbol{\Lambda} \succeq \mathbf{0},$$

where we introduced a matrix $\boldsymbol{\Lambda}$ of dual variables. The complementarity conditions for the optimal solution $(\mathbf{w}^*, t^*)$ read $\mathbf{A}((\mathbf{w}^*, t^*)) \boldsymbol{\Lambda}^* = \mathbf{0}$. The dual formulation of (6) with matrix (7) combined with the $\mathbf{F}(\mathbf{w}, t)$ part of the complementarity conditions reads

$$\underset{(\boldsymbol{\alpha}, \boldsymbol{\beta}, \boldsymbol{\gamma}) \in \mathbb{R}^{3m}}{\text{maximise}} \quad -\frac{1}{2} \sum_{i=1}^{m} \sum_{j=1}^{m} y_i y_j \left[ \tilde{\mathbf{x}}(\alpha_i, \beta_i, \gamma_i, \mathbf{X}_i) \right]^\top \left[ \tilde{\mathbf{x}}(\alpha_j, \beta_j, \gamma_j, \mathbf{X}_j) \right] + \sum_{i=1}^{m} \alpha_i$$

$$\text{subject to} \quad \forall i \in \{1, \ldots, m\}: \ \mathbf{M}_i := \left( \begin{array}{cc} \alpha_i & \beta_i \\ \beta_i & \gamma_i \end{array} \right) \succeq \mathbf{0}, \quad (8)$$

where we define extrapolated training examples $\tilde{\mathbf{x}}(\alpha_i, \beta_i, \gamma_i, \mathbf{X}_i) := \alpha_i(\mathbf{X}_i)_{0,\cdot} + \beta_i(\mathbf{X}_i)_{1,\cdot} + \gamma_i(\mathbf{X}_i)_{2,\cdot}$. As before this program with quadratic objective and psd constraints can be formulated as a standard SDP in the form (3) and is easily solved by a standard SDP solver[3]. In addition, the complementarity conditions reveal that the optimal weight vector $\mathbf{w}^*$ can be expanded as

$$\mathbf{w}^* = \sum_{i=1}^{m} y_i \tilde{\mathbf{x}}\left(\alpha_i, \beta_i, \gamma_i, \mathbf{X}_i\right), \tag{9}$$

in analogy to the corresponding result for support vector machines [2].

It remains to analyse the complementarity conditions related to the example-related $\mathbf{G}\left(\mathbf{w}, \mathbf{X}_i, y_i\right)$ constraints in (6). Using (7) and assuming primal and dual feasibility we obtain for all $i \in \{1, \ldots, m\}$ at the solution $(\mathbf{w}^*, t^*, \mathbf{M}_i^*)$,

$$\mathbf{G}\left(\mathbf{w}^*, \mathbf{X}_i, y_i\right) \cdot \mathbf{M}_i^* = \mathbf{0}, \tag{10}$$

the trace of which translates into

$$y_i \mathbf{w}^{*,\top} \left[\alpha_i^*(\mathbf{X}_i)_{0,\cdot} + \beta_i^*(\mathbf{X}_i)_{1,\cdot} + \gamma_i^*(\mathbf{X}_i)_{2,\cdot}\right] = \alpha_i^*. \tag{11}$$

These relations enable us to characterise the solution by the following propositions:

**Proposition 2 (Sparse Expansion).** *The expansion (9) of $\mathbf{w}^*$ in terms of $\mathbf{X}_i$ is sparse: Only those examples $\mathbf{X}_i$ ("support vectors") may have non-zero expansion coefficients $\alpha_i^*$ which lie on the margin, i.e., for which $\det\left(\mathbf{G}_i\left(\mathbf{w}^*, \mathbf{X}_i, y_i\right)\right) = 0$. Furthermore, in this case $\alpha_i^* = 0$ implies $\beta_i^* = \gamma_i^* = 0$.*

*Proof.* We assume $\alpha_i^* \neq 0$ and derive a contradiction. From $\mathbf{G}\left(\mathbf{w}^*, \mathbf{X}_i, y_i\right) \succ \mathbf{0}$ we conclude using Proposition 1 that for all $\theta \in \mathbb{R}$ we have $y_i \mathbf{w}^{*,\top}\left((\mathbf{X}_i)_{0,\cdot} + \theta(\mathbf{X}_i)_{1,\cdot} + \theta^2(\mathbf{X}_i)_{2,\cdot}\right) > 1$. Furthermore, we conclude from (10) that $\det(\mathbf{M}_i^*) = \alpha_i^* \gamma_i^* - \beta_i^{*2} = 0$, which together with the assumption $\alpha_i^* \neq 0$ implies that there exists $\tilde{\theta} \in \mathbb{R}$ such that $\beta_i^* = \tilde{\theta}\alpha_i^*$ and $\gamma_i^* = \beta_i^{*2}/\alpha_i^* = \tilde{\theta}^2\alpha_i^*$. Inserting this into (11) leads to a contradiction, hence $\alpha_i^* = 0$. Then, $\det(\mathbf{M}_i^*) = 0$ implies $\beta_i^* = 0$ and the fact that $\mathbf{G}\left(\mathbf{w}^*, \mathbf{X}_i, y_i\right) \succ \mathbf{0} \Rightarrow y_i \mathbf{w}^{*,\top}(\mathbf{X}_i)_{2,\cdot} \neq 0$ ensures that $\gamma_i^* = 0$ holds as well. $\square$

**Proposition 3 (Truly Virtual Support Vectors).** *For all examples $\mathbf{X}_i$ lying on the margin, i.e., satisfying $\det\left(\mathbf{G}\left(\mathbf{w}^*, \mathbf{X}_i, y_i\right)\right) = 0$ and $\det\left(\mathbf{M}_i^*\right) = 0$ there exist $\theta_i \in \mathbb{R} \cup \{\infty\}$ such that the optimal weight vector $\mathbf{w}^*$ can be written as*

$$\mathbf{w}^* = \sum_{i=1}^{m} \alpha_i^* y_i \tilde{\mathbf{x}}_i\left(\theta_i\right) = \sum_{i=1}^{m} y_i \alpha_i^* \left((\mathbf{X}_i)_{0,\cdot} + \theta_i^*(\mathbf{X}_i)_{1,\cdot} + \theta_i^{*2}(\mathbf{X}_i)_{2,\cdot}\right)$$

*Proof.* (sketch) We have $\det(\mathbf{M}_i^*) = \alpha^*\gamma^* - \beta^{*2} = 0$. We only need to consider $\alpha_i^* \neq 0$, in which case there exists $\theta_i^*$ such that $\beta_i^* = \theta_i^*\alpha_i^*$ and $\gamma_i^* = \theta_i^{*2}\alpha_i^*$. The other cases are ruled out by the complementarity conditions (10). $\square$

Based on this proposition it is possible not only to identify which examples $\mathbf{X}_i$ are used in the expansion of the optimal weight vector $\mathbf{w}^*$, but also the corresponding values $\theta_i^*$ of the transformation parameter $\theta$. This extends the idea of virtual support vectors [7] in that Semidefinite Programming Machines are capable of finding truly virtual support vectors that were not explicitly provided in the training sample.

# 3 Extensions to SDPMs

**Optimisation on a Segment** In many applications it may not be desirable to enforce correct classification on the entire trajectory given by the polynomial example $\tilde{\mathbf{x}}(\theta)$. In particular, when the polynomial is used as a local approximation to a global invariance we would like to restrict the example to a segment of the trajectory. To this end consider the following corollary to Proposition 1.

**Corollary 1 (SD-Representability on a segment [5]).** *For any $l \in \mathbb{N}$, the set $\mathcal{P}_l^+ (-\tau, \tau)$ of polynomials non-negative on a segment $[-\tau, \tau]$ is SD-representable.*

*Proof.* (sketch) Consider a polynomial $p \in \mathcal{P}_l^+ (-\tau, \tau)$ where $p := x \mapsto \sum_{i=0}^l p_i x^i$ and

$$q := x \mapsto \left(1 + x^2\right)^l \cdot \left[p(\tau(2x^2(1+x^2)^{-1} - 1))\right].$$

If $q \in \mathcal{P}_{2l}^+$ is non-negative everywhere then $p$ is non-negative in $[-\tau, \tau]$. $\qquad\square$

The proposition shows how we can restrict the examples $\tilde{\mathbf{x}}(\theta)$ to a segment $\theta \in [-\tau, \tau]$ by effectively doubling the degree of the polynomial used. This is the SDPM version used in the experiments in Section 4. Note that the matrix $\mathbf{G}(\mathbf{w}, \mathbf{X}_i, y_i)$ is sparse because the resulting polynomial contains only even powers of $\theta$.

**Multiple Transformation Parameters** In practice it would be desirable to treat more than one transformation at once. For example, in handwritten digit recognition transformations like rotation, scaling, translation, shearing, thinning/thickening etc. may all be relevant [8]. Unfortunately, Proposition 1 only holds for polynomials in one variable. However, its first statement may be generalised to polynomials of more than one variable: for every psd matrix $\mathbf{P} \succeq \mathbf{0}$ the polynomial $p(\boldsymbol{\rho}) = \boldsymbol{\theta}_{\boldsymbol{\rho}}^\top \mathbf{P} \boldsymbol{\theta}_{\boldsymbol{\rho}}$ is non-negative everywhere, even if $\theta_i$ is any monomial in $\rho_1, \ldots, \rho_D$. This means, that optimisation is only over a subset of these polynomials[4]. Considering polynomials of degree two and $\boldsymbol{\theta}_{\boldsymbol{\rho}} := (1, \rho_1, \ldots, \rho_D)$ we have,

$$\tilde{x}_i(\boldsymbol{\rho}) \approx \boldsymbol{\theta}_{\boldsymbol{\rho}}^\top \begin{bmatrix} x_i(\mathbf{0}) & \nabla_{\boldsymbol{\rho}}^\top x_i(\mathbf{0}) \\ \nabla_{\boldsymbol{\rho}} x_i(\mathbf{0}) & \nabla_{\boldsymbol{\rho}} \nabla_{\boldsymbol{\rho}}^\top x_i(\mathbf{0}) \end{bmatrix} \boldsymbol{\theta}_{\boldsymbol{\rho}},$$

where $\nabla_{\boldsymbol{\rho}}^\top$ denotes the gradient and $\nabla_{\boldsymbol{\rho}} \nabla_{\boldsymbol{\rho}}^\top$ denotes the Hessian operator.

Note that the scaling behaviour with regard to the number $D$ of parameters is more benign than that of the naive method of adding virtual examples to the training sample on a grid. Such a procedure would incur an exponential growth in the number of examples, whereas the approximation above only exhibits a linear growth in the size of the matrices involved.

**Learning with Kernels** Support vector machines derive much of their popularity from the flexibility added by the use of kernels [2, 7]. Due to space restrictions we cannot discuss kernels in detail. However, taking the dual SDPM (8) as a starting point and assuming the Taylor expansion (2) the crucial point is that in order to represent the polynomial trajectory in feature space we need to differentiate through the kernel function.

Let us assume a feature map $\boldsymbol{\phi} : \mathbb{R}^n \to \mathcal{F} \subseteq \mathbb{R}^N$ and $k : \mathcal{X} \times \mathcal{X} \to \mathbb{R}$ be the kernel function corresponding to $\boldsymbol{\phi}$ in the sense that $\forall \mathbf{x}, \tilde{\mathbf{x}} \in \mathcal{X} : [\boldsymbol{\phi}(\mathbf{x})]^\top [\boldsymbol{\phi}(\tilde{\mathbf{x}})] = k(\mathbf{x}, \tilde{\mathbf{x}})$.

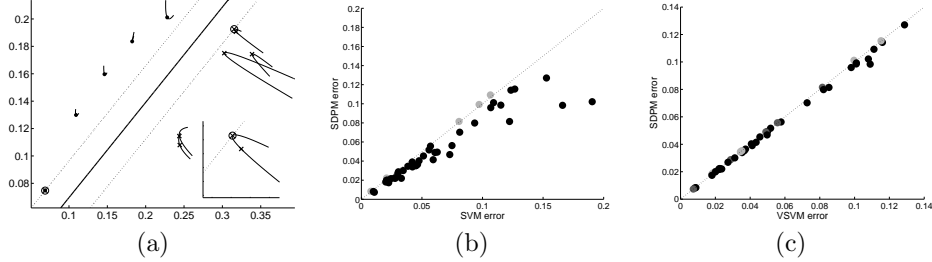

$$\qquad\qquad \text{(a)} \qquad\qquad\qquad\qquad \text{(b)} \qquad\qquad\qquad\qquad \text{(c)}$$

Figure 2: (**a**) A linear classifier learned with the SDPM on 10 2D-representations of the USPS digits "1" and "9" (see Figure 1 for details). Note that the "support" vector is truly virtual since it was never directly supplied to the algorithm (inset zoom-in). (**b**) Mean test errors of classifiers learned with the SVM vs. SDPM (see text) and (c) virtual SVM vs. SDPM algorithm on 50 independent training sets of size $m = 20$ for all 45 digit classification tasks.

The Taylor expansion (2) is now carried out in $\mathcal{F}$. Then an inner product expression between data points $\mathbf{x}_i$ and $\mathbf{x}_j$ differentiated, respectively, $u$ and $v$ times reads

$$\underbrace{\left[ \boldsymbol{\phi}^{(u)}(\mathbf{x}_i)\right]^{\top} \left[ \boldsymbol{\phi}^{(v)}(\mathbf{x}_j)\right]}_{k^{(u,v)}(\mathbf{x}_i,\mathbf{x}_j)} = \sum_{s=1}^{N} \left( \left. \frac{d^u \phi_s(\mathbf{x}(\theta))}{d\theta^u}\right|_{\mathbf{x}=\mathbf{x}_i,\theta=0}\right) \cdot \left( \left. \frac{d^v \phi_s(\tilde{\mathbf{x}}(\tilde{\theta}))}{d\tilde{\theta}^v}\right|_{\tilde{\mathbf{x}}=\mathbf{x}_j,\tilde{\theta}=0}\right) \; .$$

The kernel trick may help avoid the sum over $N$ feature space dimensions, however, it does so at the cost of additional terms by the product rule of differentiation. It turns out that for polynomials of degree $r = 2$ the exact calculation of elements of the kernel matrix is already $\mathcal{O}\left(n^4\right)$ and needs to be approximated efficiently in practice.

## 4   Experimental Results

In order to test and illustrate the SDPM we used the well-known USPS data set of $16 \times 16$ pixel images in $[0, 1]$ of handwritten digits. We considered the transformation *rotation by angle* $\theta$ and calculated the first and second derivatives $\mathbf{x}_i'\,(\theta = 0)$ and $\mathbf{x}_i''\,(\theta = 0)$ based on an image representation smoothed by a Gaussian of variance 0.09.

For the purpose of illustration we calculated two simple features, averaging the first and the second 128 pixel intensities, respectively. Figure 2 (a) shows a plot of 10 training examples of digits "1" and "9" together with the quadratically approximated trajectories for $\theta \in [-20°, 20°]$. The examples are separated by the solution found with an SDPM restricted to the same segment of the trajectory. Following Propositions 2 and 3 the weight vector found is expressed as a linear combination of truly virtual support vectors that had not been supplied in the training sample directly (see inset).

In a second experiment, we probed the performance of the SDPM algorithm on the full feature set of 256 pixel intensities using 50 training sets of size $m = 20$ for each of the 45 one-versus-one classification tasks between all of the digits from "0" to "9" from the USPS data set. For each task, the digits in one class were rotated by $-10°$ and the digits of the other class by $+10°$. We compared the performance of the SDPM algorithm to the performance of the original support vector machine (SVM) [2] and the virtual support vector machine (VSVM) [7] measured on independent test sets of size 250. The VSVM takes the support vectors of the ordinary SVM run and is trained on a sample that contains these support vectors together with transformed versions rotated by $-10°$ and $+10°$ in the quadratic approximation. The results are

shown in the form of scatter plots of the errors for the 45 tasks in Figure 2 (b) and (c). Clearly, taking into account the invariance is useful and leads to SDPM performance superior to the ordinary SVM. The SDPM also performs slightly better than the VSVM, however, this could be attributed to the pre-selection of support vectors to which the transformation is applied. It is expected that for increasing number $D$ of transformations the performance improvement becomes more pronounced because in high dimensions most volume is concentrated on the boundary of the convex hull of the polynomial manifold.

## 5   Conclusion

We introduced Semidefinite Programming Machines as a means of learning on infinite families of examples given in terms of polynomial trajectories or—more generally— manifolds in data space. The crucial insight lies in the SD-representability of non-negative polynomials which allows us to replace the simple non-negativity constraint in algorithms such as support vector machines by positive semidefinite constraints.

While we have demonstrated the performance of the SDPM only on very small data sets it is expected that modern interior-point methods make it possible to scale SDPMs to problems of $m \approx 10^5 - 10^6$ data points, in particular in primal space where the number of variables is given by the number of features. This expectation is further supported by the following: (i) The resulting SDP is well structured in the sense that $\mathbf{A}(\mathbf{w}, t)$ is block-diagonal with many small blocks. (ii) It may often be sufficient to satisfy the constraints—e.g., by a version of the perceptron algorithm for semidefinite feasibility problems [3]—without necessarily maximising the margin.

Open questions remain about training SDPMs with multiple parameters and about the efficient application of SDPMs with kernels. Finally, it would be interesting to obtain learning theoretical results regarding the fact that SDPMs effectively make use of an infinite number of (non IID) training examples.

## Footnotes

[1] We omit an explicit threshold to unclutter the presentation.

[2] The characteristic polynomial of a $2 \times 2$ matrix is quadratic and has at most two solutions. The condition that the lower eigenvalue be non-negative can be expressed as a second-order cone constraint. The SOCP formulation—if applicable—can be solved more efficiently than the SDP formulation.

[3]We used the SDP solver SeDuMi together with the LMI parser Yalmip under MATLAB (see also *http://www-user.tu-chemnitz.de/˜helmberg/semidef.html*).

[4]There exist polynomials in more than one variable that are non-negative everywhere yet cannot be written as a sum of squares and are hence not SD-representable.

## References

[1] O. Chapelle and B. Schölkopf. Incorporating invariances in non-linear support vector machines. In T. G. Dietterich, S. Becker, and Z. Ghahramani, editors, *Advances in Neural Information Processing Systems 14*, pages 609–616, Cambridge, MA, 2002. MIT Press.

[2] C. Cortes and V. Vapnik. Support vector networks. *Machine Learning*, 20:273–297, 1995.

[3] T. Graepel, R. Herbrich, A. Kharechko, and J. Shawe-Taylor. Semidefinite programming by perceptron learning. In S. Thrun, L. Saul, and B. Schölkopf, editors, *Advances in Neural Information Processing Systems 16*. MIT Press, 2004.

[4] A. Nemirovski. Five lectures on modern convex optimization, 2002. Lecture notes of the C.O.R.E. Summer School on Modern Convex Optimization.

[5] Y. Nesterov. Squared functional systems and optimization problems. In H. Frenk, K. Roos, T. Terlaky, and S. Zhang, editors, *High Performance Optimization*, pages 405– 440. Kluwer Academic Press, 2000.

[6] F. Rosenblatt. The perceptron: A probabilistic model for information storage and organization in the brain. *Psychological Review*, 65(6):386–408, 1958.

[7] B. Schölkopf. *Support Vector Learning*. R. Oldenbourg Verlag, München, 1997. Doktorarbeit, TU Berlin. Download: http://www.kernel-machines.org.

[8] P. Simard, Y. LeCun, J. Denker, and B. Victorri. Transformation invariance in pattern recognition, tangent distance and tangent propagation. In G. Orr and M. K., editors, *Neural Networks: Tricks of the trade*. Springer, 1998.

[9] L. Vandenberghe and S. Boyd. Semidefinite programming. *SIAM Review*, 38(1):49–95, 1996.
